# Size Matters: Metric Visual Search Constraints from Monocular Metadata

**Mario Fritz**
UC Berkeley EECS & ICSI

**Kate Saenko**
UC Berkeley EECS & ICSI

**Trevor Darrell**
UC Berkeley EECS & ICSI

## Abstract

Metric constraints are known to be highly discriminative for many objects, but if training is limited to data captured from a particular 3-D sensor the quantity of training data may be severly limited. In this paper, we show how a crucial aspect of 3-D information–object and feature *absolute size*–can be added to models learned from commonly available online imagery, without use of any 3-D sensing or reconstruction at training time. Such models can be utilized at test time together with explicit 3-D sensing to perform robust search. Our model uses a "2.1D" local feature, which combines traditional appearance gradient statistics with an estimate of average absolute depth within the local window. We show how category size information can be obtained from online images by exploiting relatively unbiquitous metadata fields specifying camera intrinsics. We develop an efficient metric branch-and-bound algorithm for our search task, imposing 3-D size constraints as part of an optimal search for a set of features which indicate the presence of a category. Experiments on test scenes captured with a traditional stereo rig are shown, exploiting training data from from purely monocular sources with associated EXIF metadata.

## 1 Introduction

Two themes dominate recent progress towards situated visual object recognition. Most significantly, the availability of large scale image databases and machine learning methods has driven performance: accuracy on many category detection tasks is a function of the *quantity and quality of the available training data*. At the same time, when we consider situated recognition tasks, i.e., as performed by robots, autonomous vehicles, and interactive physical devices (e.g., mobile phones), it is apparent that the *variety and number of sensors* is often what determines performance levels: e.g., the avaibility of 3-D sensing can significantly improve performance on specific practical tasks, irrespective of the amount of training data. A rich variety of 3-D sensors are available on modern robotic systems, yet the training data are few for most 3-D sensor regimes: the vast majority of available online visual category data are from monocular sources and there are few databases of real-world 3-D scans from which to train robust visual recognizers. In general it is, however, difficult to reconcile these two trends: while one would like to use all available sensors at test time, the paucity of 3D training data will mean few categories are well-defined with full 3-D models, and generalization performance to new categories which lack 3-D training data may be poor. In this paper, we propose a method to bridge this gap and extract features from typical 2D data sources that can enhance recognition performance when 3D information is available at test time.

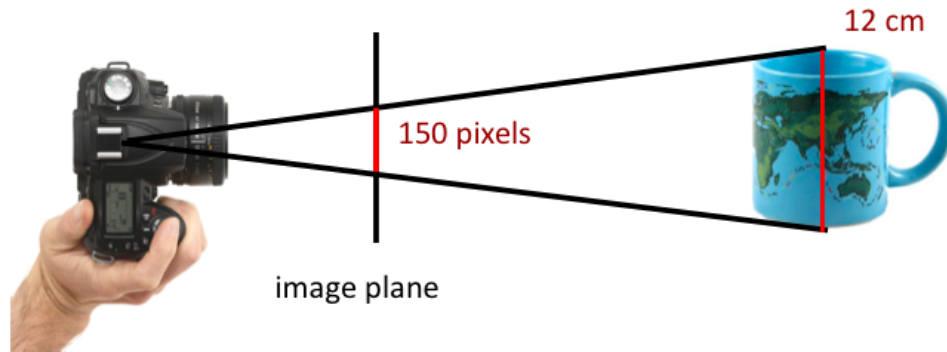

Figure 1: Recovery of object size from known camera intrinsics

The paradigm of recognition-by-local-features has been well established in the computer vision literature in recent years. Existing recognition schemes are designed generally to be invariant to scale and size. Local shape descriptors based on 3-D sensing have been proposed (e.g., VIP [2]), as well as local 3-D descriptors (e.g., 3-D shape context and SIFT [4, 3]), but we are somewhat skeptical of the ability of even the most recent 3-D sensor systems to extract the detailed local geometry required to reliably detect and describe local 3-D shapes on real world objects.

Instead of extracting full 3D local features, we propose a "2.1D" local feature model which augments a traditional 2D local feature (SIFT, GLOH, SURF, etc.) with an estimate of the depth and 3-D size of an observed patch. Such features could distinguish, for example, the two different keypad patterns on a mobile device keyboard vs. on a full-size computer keyboard; while the keys might look locally similar, the absolute patch size would be highly distinctive. We focus on the recognition of real-world objects when additional sensors are available at test time, and show how 2.1D information can be extracted from monocular metadata already present in many online images. Our model includes both a representation of the absolute size of local features, and of the overall dimension of categories. We recover the depth and size of the local features, and thus of the bounding box of a detected object in 3-D. Efficient search is an important goal, and we show a novel extension to multi-class branch-and-bound search using explicit metric 3-D constraints.

## 2 Recognition with "2.1D" features

The crux of our method is the inference and exploitation of size information; we show that we can obtain such measurements from non-traditional sources that do not presume a 3-D scanner at training time, nor rely on multi-view reconstruction / structure-from-motion methods. We instead exploit cues that are readily available in many monocular camera images.[1] We are not interested in reconstructing the object surface, and only estimate the absolute size of local patches, and the statistics of the bounding box of instances in the category; from these quantities we can infer the category size.

We adopt a local-feature based recognition model and augment it with metric size information. While there are several possible local feature recognition schemes based on sets of such local features, we focus on the Naive Bayes nearest-neighbor model of [1] because of its simplicity and good empirical results. We assume one or more common local feature descriptors (and associated detectors or dense sampling grids): SIFT, SURF, GLOH, MSER. Our emphasis in this paper is on

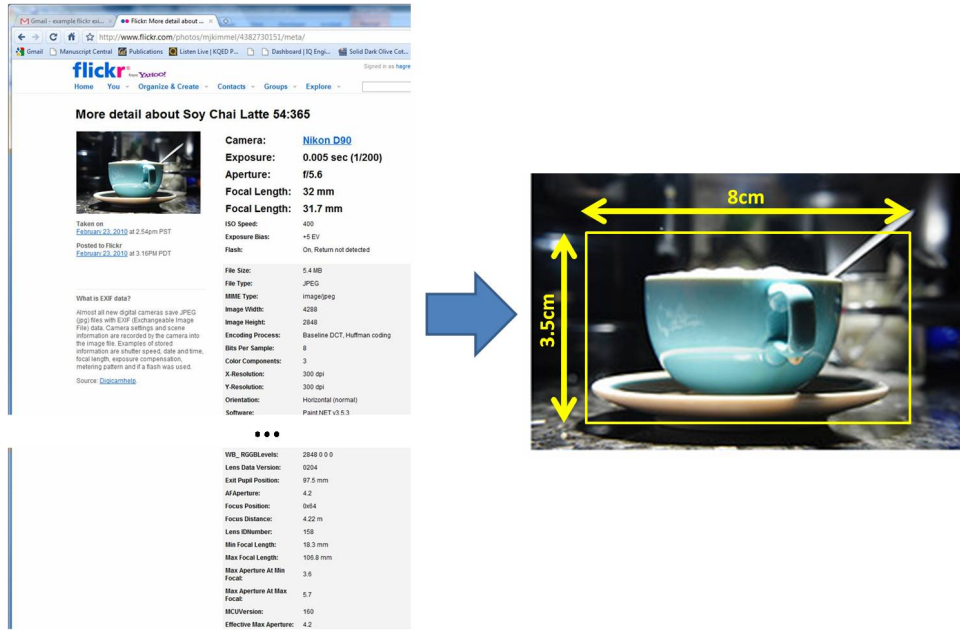

Figure 2: Illustration of metric object size derived from image metadata stored in EXIF fields on an image downloaded from Flickr.com. Absolute size is estimated by projecting bounding box of local features on object into 3-D using EXIF camera intrinsics stored in image file format.

improving the accuracy of recognizing categories that are at least approximately well modeled with such-local feature schemes; size information alone cannot help recognize a category that does not repeatably and reliably produce such features.

## 2.1 Metric object size from monocular metadata

Absolute pixel size can be infered using a planar object approximation and depth from focus cues. Today's digital cameras supplement the image data with rich meta-data provided in the EXIF format. EXIF stores a wide range of intrinsic camera parameters, which often include the focus distance as an explicit parameter (in some cameras it is not provided directly, but can be estimated from other provided parameters). This gives us a workable approximation of the depth of the object, assuming it is in focus in the scene: with a pinhole camera model, we can derive the metric size of a pixel in the scene given these assumptions. Using simple trigonometry, the metric pixel size is $\rho = \frac{sd}{fr}$, where $s$ is the sensor width, $d$ is the focus distance, $f$ is the focal length, and $r$ is the horizontal resolution of the sensor.

As shown in Figure 2, this method provides a size estimate reference for the visual observation based on images commonly available on the internet, e.g., Flickr.com. A bounding box can either be estimated from the feature locations, given an uncluttered background, or provided by manual labeling or by an object discovery technique which clusters local features to discover the segmentation of the training data.

## 2.2 Naive Bayes estimation of discriminative feature weights

Our object model is based on a bag-of-words model where an object is encoded by a set of visual features $x_i \in X$ within the circumscribing bounding box. Our size-constrained learning scheme is applicable to a range of recognition methods; for simplicity we adopt a simple but efficient non-parametric naive Bayes scheme. We denote object appearance with $p(X|C)$; following [1], this density can be captured and modeled using Parzen window density estimates:

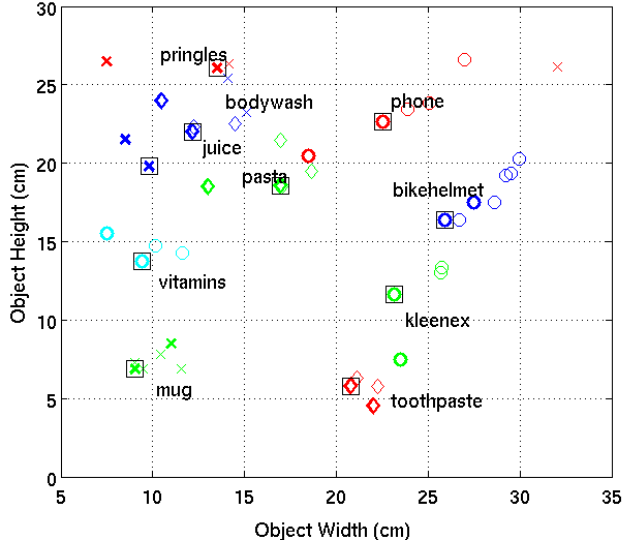

Figure 3: Metric object size for ten different categories derived from camera metadata. Bold symbols depict ground truth obtained by direct physical measurement of category instance.

$$\hat{p}(x|C) = \frac{1}{N} \sum_{j=1}^{N} K(x - x_{\mathrm{j}}^{C}), \tag{1}$$

where $K(.)$ is a Gaussian kernel.

We extend this model in a discriminative fashion similar to [18]. We compute the detection score for a given bounding box from the log-likelihood ratio computed based on the kernel density estimate from above. Assuming independence of the features, the class specfic probabilities are factorized to obtain a sum of individual feature contributions:

$$\log \frac{p(X|C)}{p(X|\bar{C})} = \log \frac{\prod_i p(x_i|C)}{\prod_i p(x_i|\bar{C})} \tag{2}$$

$$= \sum_i \log(p(x_i|C) - log(p(x_i|\bar{C})) \tag{3}$$

As shown in [1], an approximate density based only on the nearest neighbor is accurate for many recognition tasks. This further simplifies the computation and approximates the class specific feature probabilities by:

$$\log p(x_i|C) \approx ||x_i - NN_C(x_i)||^2, \tag{4}$$

where $NN_C(x_i)$ represents the distance of data point $x_i$ to the nearest example in the training data of class $C$. In the multi-class case, each feature $x_i$ is compared to the nearest neighbors in the training examples of each class, $NN_{\bar{C}}(x_i)$ can be simply obtained as the minimum of all retrieved nearest neighbors except those in C.

## 3 Efficient search with absolute size

Recently, a class of algorithms for efficient detection based on local features has been proposed [19, 20, 21]; these search for the highest-scoring bounding box given the observed features $X$ and a

scoring function $f$ using an efficient branch-and-bound scheme. These methods can be formulated as an optimization $b = \arg\max_b f(b)$, where $b = (x_1, y_1, x_2, y_2)$ is a bounding box. The core idea is to structure the search space using a search tree. The top node contains the set of all possible bounding boxes. The child nodes contain splits of the set of bounding boxes in the parent node. The leafs contain single bounding boxes. If it is possible to derive lower and upper bounds for rectangle sets at the nodes, a branch and bound technique can be applied to quickly prune nodes if its upper bound is lower than the lower bound of a previously visited node.

Bounds can be easily computed for bag-of-words representations, which have been previously used in this context for object detection. Each feature has a learned weight $w_j$, wherefore the score function $f$ reads:

$$f(r) = \sum_{j \in T(b)} w_j, \tag{5}$$

where $T(b)$ is the set of all features contained in the bounding box $b$.

While previous approaches have derived the feature weight from SVM training, we propose to use likelihood ratios which are derived in a non-parametric fashion.

We further extend this method to search for objects in 3d. Our bounding box hypotheses $b = (x_1, y_1, z_1, x_2, y_2, z_2)$ are defined explicitly in 3d and indicate the actual spatial relation of objects in the scene.

We employ a constraint factor $S(b)$ to the objective that indicates if a bounding box has a valid size given a particular class or not:

$$f(r) = \sum_{j \in T(b)} w_j S(b) \tag{6}$$

$S(b) = 1$ is a basic rectangle function that takes the value 1 for valid bounding boxes and 0 otherwise.

Most importantly, bounds over bounding box sets can still be efficiently computed. As long as the bounding box set at a given node in the search tree contains at least one bounding box of valid size, the score is unaffected. When there is no valid rectangle left, the score evaluates to zero and that node as well as the associated sub space of the search problem gets pruned.

At test time, it is anticipated that 3D observations are directly available via LIDAR scans or active or passive stereo estimation. Given these measurements, we constrain the search to leverage the metric information acquired at training time. The depth for each feature in the image at test time allows us to infer their 3D location in the test scene. We can thus extend efficient multi-class branch-and-bound search to operate in metric 3D space under the constraints imposed by our knowledge of metric patch size and metric object size.

We also make use of the proposed multi-class branch-and-bound scheme as proposed in [20]. We not only split bounding box sets along dimensions, but also split the set of object classes. This leads to a simultaneous search scheme for multiple classes.

## 4 Related Work

Many methods have been proposed to deal with the problem of establishing feature correspondence across varying image scales. Lowe et. al. proposed to up/downsample an image at multiple scales and identify the characterstic scale for each image patch [9]. A histogram of edge orientations is computed for each patch scaled to its characteristic scale in order to obtain a scale-invariant visual descriptor. [10] identifies scale invariant regions by iteratively expanding consistent regions with an increasing intensity threshold until they become "stable". The size of the stable region is the charactersitic scale for the feature. With both methods, a feature in one image can be mapped to the same characteristic scale a feature in another image. Since both features are mapped to the same

scale, an "apple-to-apple" comparison can be performed. In contrast, our method does not require such a mapping. Instead, it determines the metric size of any image patch and uses it to compare two features directly.

There have been several works on estimating depth from single images. Some very early work estimated depth from the degree of the defocus of edges [8]. [6] describes a method to infer scene depth from structure baesd on global and local histograms of Gabor filter responses for indoor and outdoor scenes. [11] describes a supervised Markov Random Field method to predict the depth from local and global features for outdoor images. In our work, we focus on indoor office scenes with finer granularity. Hardware-based methods for obtaining 3D information from monocular images include modifying the structure of a conventional camera to enable it to capture 3D geometry. For example, [12] introduces the coded aperture technique by inserting a patterned occluder within the aperture of the camera lenses. Images captured by such a camera exhibit depth-dependent patterns from which a layered depth map can be extracted.

Most methods based on visual feature quantization learn their codebooks using invariant features. However, the scale of each code word is lost after each image patch is normalized to its invariant region. Thus, it is possible for two features to match because they happen to look similar, even though in the physical world they actually have two different sizes. For example, an eye of a dinosaur may be confused with an eye of a fish, because their size difference is lost once they are embedded into the visual code book. There have been some proposals to deal with this problem. For example, [13] records the relative position of the object center in the codebook, and at test time each codebook word votes for the possible object center at multiple scales. Moreover, [14] explicitly put the orientation and scale of each feature in the codebook, so that object center location can be inferred directly. However, these works treat orientation and scale as independent of the feature descriptor and use them to post-verify whether a feature found to be consistent in terms of the appearance desciptor would also be consistent in terms of scale. In contrast, our work directly embeds the scale attribute into the visual descriptor. A visual word would be matched only if its size is right. In other words, the visual apperance and the scale are matched simulaneously in our codebook.

Depth information has been used to improve the performance of various image processing tasks, such as video retrieval, object instance detection, 3D scene recognition, and vehicle navigation. For example, [15] used depth feature for video retrieval, extracting depth from monocular video sequences by exploiting the motion parallax of the objects in the video. [16] developed an intergrated probablistic model for apperance and 3D geometry of object categories. However, their method does not expliclty assign physical size to each image patch and needs to provide scale-invariance by explictly calculating the perspective projection of objects in different 3D poposes. In contrast, our method can infer the real-world sizes of features and can establish feature correspondences at their true physical scale. [17] proposed a way to use depth estimation for real-time obstacle detection from a monocular video stream in a vehicle navigation scenario. Their method estimates scene depth from the scaling of supervised image regions and generates obstacle hypotheses from these depth estimates.

## 5 Experiments

In the experiments we show how to improve performance of visual object classifiers by leveraging richer sensor modalities deployed at test time. We analyze how the different proposed means of putting visual recognition in metric context improves detection performance.

### 5.1 Data

For training we explore the camera-based metadata scheme described above, where we derive the metric pixel size from EXIF data. We downloaded 38 images of 10 object categories taken with a consumer grade dSLR that stores relevant EXIF fields (e.g., Nikon D90). For test data we have collected 34 scenes in our laboratory of varying complexity containing 120 object instances in offices and and a kitchen. Considerable levels of clutter, lighting and occlusion are present in the test set. Stereo depth observations using a calibrated camera rig are obtained with test imagery, providing an estimate of the 3-D depth of each feature point at test time.

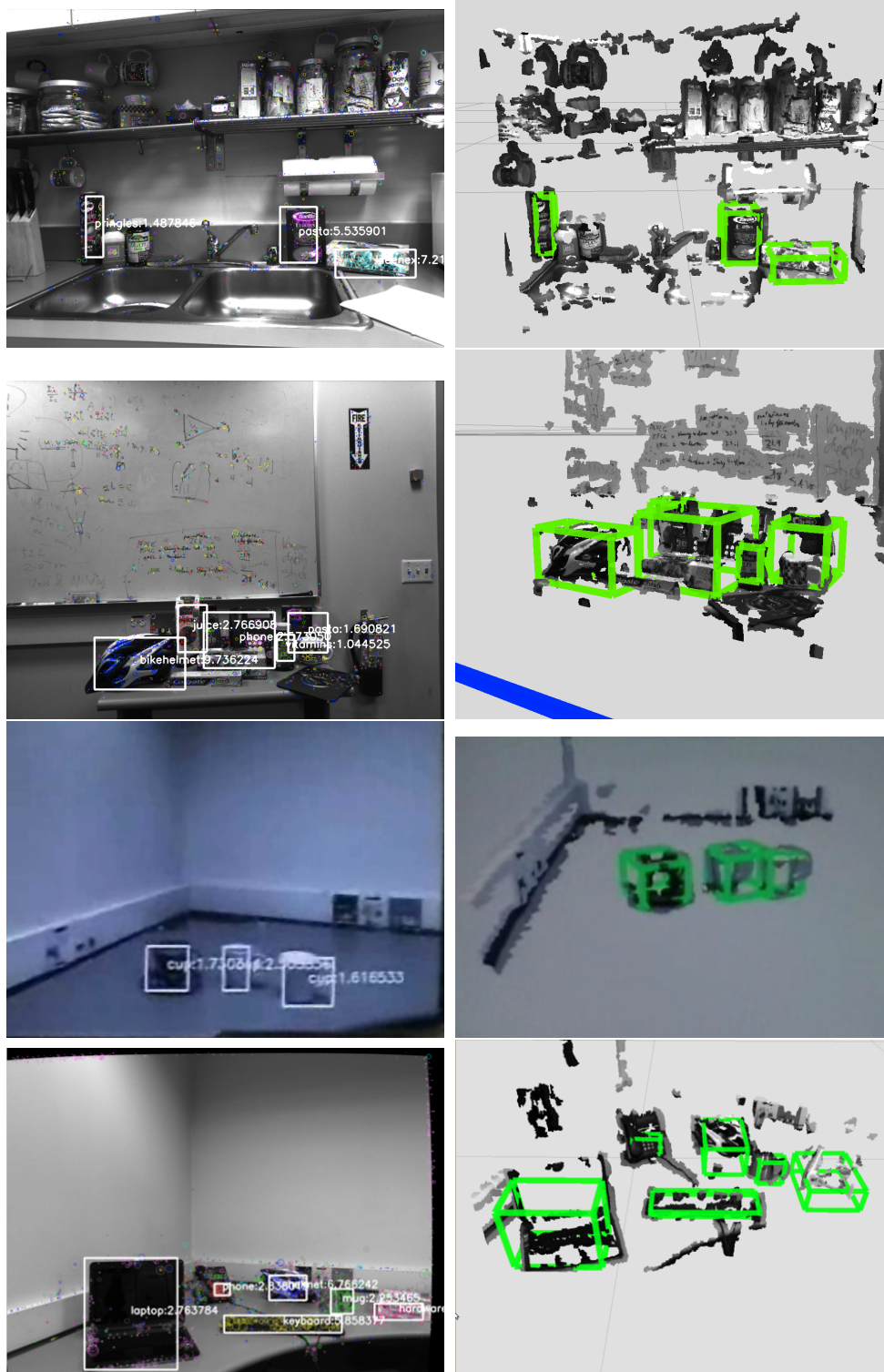

Figure 4: Example detections.

| object | baseline | 2.1D |
|---|---|---|
| bike helmet | 89.0 | 99.1 |
| body wash | 3.3 | 80.0 |
| juice | 76.0 | 100.0 |
| kleenex | 60.0 | 76.53 |
| mug | 0.0 | 24.63 |
| pasta | 36.3 | 65.6 |
| phone | 80.0 | 65.7 |
| pringles | 45.8 | 94.3 |
| toothpaste | 20.0 | 100.0 |
| vitamins | 0.0 | 60.0 |
| average | 41.0 | 76.59 |

Table 1: Average precision for several categories for baseline 2-D branch and bound search and our 2.1D method.

## 5.2 Evaluation

We start with a baseline, which uses the plain branch and bound detection scheme and 2D features. We then experiment with augment the representation to 2.1D, adding 3D location to the interest points, as well as employing the metric size constraint.

Table 1 shows the average precision for each category for baseline 2-D branch and bound search and our 2.1D method. Adding the metric object constraints (second column) improves the results significantly. As illustrated in Figure 4, our 2.1D representation allows grouping in 3-D and provides improved occlusion handling. We see that the baseline branch-and-bound performs poorly on this data set and is not capable of localizing two of the items at all. For the training data available for these categories the local evidence was apparently not strong enough to support this detection scheme, but with size constraints performance improved significantly.

## 6 Conclusion

Progress on large scale systems for visual categorization has been driven by the abundance of training data available from the web. Much richer and potentially more discriminative measurements can be acquired and leveraged by additional sensor modalities, e.g. 3D measurements from stereo or lidar, typically found on contemporary robotic platforms, but there is rarely sufficient training data to learn robust models using these sensors. In order to reconcile these two trends, we developed a method for appearance-based visual recognition in metric context, exploiting camera-based metadata to obtain size information regarding a category and local feature models that can be exploited using 3-D sensors at test time.

We believe that "size matters", and that the most informative and robust aspect of 3-D information is dimensional. We augmented local feature-based visual models with a "2.1D" object representation by introducing the notion of a metric patch size. Scene context from 3-D sensing and category-level dimension estimates provide additional cues to limit search. We presented a fast, multi-class detection scheme based on a metric branch-and-bound formulation. While our method was demonstrated only on simple 2-D SURF features, we belive these methods will be applicable as well to multi-kernel schemes with additional feature modalities, as well as object level desriptors (e.g., HOG, LatentSVM).

**Acknowledgements.** This work was supported in part by TOYOTA and a Feodor Lynen Fellowship granted by the Alexander von Humboldt Foundation.

## Footnotes

[1]There are a number of general paradigms by which estimates of object size can be extracted from a 2D image data source, e.g., regression from scene context [6]), or inference of depth-from-a-single-image [7, 11, 16]. In addition to such schemes, text associated with the training images extracted from internet merchants (e.g., Amazon, eBay) typically explicitly defines a bounding volume for the object. While all these are of interest, we consider here only the use of methods based implicitly on depth-from-focus (e.g., [8]), present as camera intrinsics stored as metadata in the JPEG EXIF file format. Images collected by many modern consumer-grade digital SLR cameras automatically store absolute distance-to-subject as metadata in the JPEG image.

# References

[1] O. Boiman, E. Shechtman, and M. Irani, *In defense of Nearest-Neighbor based image classification*, In Proceedings of Computer Vision and Pattern Recognition, 2008.

[2] C. Wu, B. Clipp, X. Li, J.-M. Frahm, and M. Pollefeys, *3D model matching with Viewpoint-Invariant Patches (VIP)*, In Proceedings of Computer Vision and Pattern Recognition, 2008.

[3] P. Scovanner, S. Ali, M. Shah, *A 3-dimensional SIFT descriptor and its application to action recognition*, In Proceedings of the 15th international conference on Multimedia, 2007.

[4] M. Kortgen, G. J. Park, M. Novotni, R. Klein, *3D Shape Matching with 3D Shape Contexts*, In the 7th Central European Seminar on Computer Graphics, 2003.

[5] A. Frome, D. Huber, R. Kolluri, T. Bulow, and J. Malik. *Recognizing objects in range data using regional point descriptors*, In Proceedings of the 8th European Conference on Computer Vision, 2004.

[6] A. Oliva, and A. Torralba, *Building the Gist of a Scene: The Role of Global Image Features in Recognition*, In Visual Perception, Progress in Brain Research, vol 155, 2006.

[7] D. Hoiem, A. Efros, M. Hebert, *Geometric Context from a Single Image*, In Proceedings of the Tenth IEEE International Conference on Computer Vision, 2005.

[8] T. Darrell and K. Wohn, *Pyramid based depth from focus*, In Proceedings of Computer Vision and Pattern Recognition, 1988.

[9] D. Lowe, *Distinctive Image Features from Scale-Invariant Keypoints*, International Journal of Computer Vision, 2004.

[10] J. Matas, O. Chum, and M. Urban, and T. Pajdla, Robust wide baseline stereo from maximally stable extremal regions. In British Machine Vision Conference, 2002.

[11] A. Saxena, M. Sun, A. Y. Ng, *Make3D: Learning 3-D Scene Structure from a Single Still Image*, In IEEE Transactions on Pattern Analysis and Machine Intelligence (PAMI), 2008.

[12] A. Levin, R. Fergus, F. Durand, Frédo, and W.T. Freeman, *Image and depth from a conventional camera with a coded aperture*, ACM Transactions on Graphics, 2007.

[13] Bastian Leibe and Ales Leonardis and Bernt Schiele, Combined Object Categorization and Segmentation With An Implicit Shape Model In ECCV workshop on statistical learning in computer vision, 2004

[14] Krystian Mikolajczyk and Cordelia Schmid, *A Performance Evaluation of Local Descriptors*, In PAMI, 2005.

[15] R. Ewerth, M. Schwalb, Martin, and B. Freisleben, *Using depth features to retrieve monocular video shots*, In Proceedings of the 6th ACM international conference on image and video retrieval, 2007.

[16] E. Sudderth, A. Torralba, W. T. Freeman, and A. Wilsky, *Depth from Familiar Objects: A Hierarchical Model for 3D Scenes*, In Proceedings of Computer Vision and Pattern Recognition, 2006.

[17] A. Wedel, U. Franke, J. Klappstein, T. Brox, and D. Cremers, *Realtime Depth Estimation and Obstacle Detection from Monocular Video*, DAGM-Symposium, 2006.

[18] Junsong Yuan, Zicheng Liu and Ying Wu, *Discriminative Subvolume Search for Efficient Action Detection*, In Proceedings of Computer Vision and Pattern Recognition, 2009.

[19] Christoph H. Lampert and Matthew B. Blaschko and Thomas Hofmann, *Efficient Subwindow Search: A Branch and Bound Framework for Object Localization*, In Transactions on Pattern Analysis and Machine Intelligence (PAMI), 2009.

[20] Tom Yeh, John Lee and Trevor Darrell, *Fast Concurrent Object Localization and Recognition*, In CVPR 2009.

[21] Junsong Yuan and Zicheng Liu and Ying Wu, *Discriminative Subvolume Search for Efficient Action Detection*, In CVPR 2009.

